# Multiple Instance Learning for Computer Aided Diagnosis

**Glenn Fung, Murat Dundar, Balaji Krishnapuram, R. Bharat Rao**
CAD & Knowledge Solutions, Siemens Medical Solutions USA, Malvern, PA 19355
{glenn.fung, murat.dundar, balaji.krishnapuram, bharat.rao}@siemens.com

## Abstract

Many *computer aided diagnosis* (CAD) problems can be best modelled as a multiple-instance learning (MIL) problem with unbalanced data: *i.e.* , the training data typically consists of a few positive bags, and a very large number of negative instances. Existing MIL algorithms are much too computationally expensive for these datasets. We describe CH, a framework for learning a Convex Hull representation of multiple instances that is significantly faster than existing MIL algorithms. Our CH framework applies to any standard hyperplane-based learning algorithm, and for some algorithms, is guaranteed to find the global optimal solution. Experimental studies on two different CAD applications further demonstrate that the proposed algorithm significantly improves diagnostic accuracy when compared to both MIL and traditional classifiers. Although not designed for standard MIL problems (which have both positive and negative bags and relatively balanced datasets), comparisons against other MIL methods on benchmark problems also indicate that the proposed method is competitive with the state-of-the-art.

## 1    Introduction

In many *computer aided diagnosis* applications, the goal is to detect potentially malignant tumors and lesions in medical images (CT scans, X-ray, MRI etc). In an almost universal paradigm for CAD algorithms, this problem is addressed by a 3 stage system: identification of potentially unhealthy regions of interest (ROI) by a candidate generator, computation of descriptive features for each candidate, and labeling of each candidate (*e.g.* as normal or diseased) by a classifier. The training dataset for the classifier is generated as follows: Expert radiologists examine a set of images to mark out tumors. Then, candidate ROIs (with associated computed features) are marked positive if they are sufficiently close to a radiologist mark, and negative otherwise. Many CAD datasets have fewer than 1-10% positive candidates. In the CAD literature, standard machine learning algorithms—such as *support vector machines* (SVM), and *Fisher's linear discriminant*—have been employed to train the classifier. In Section 2 we show that CAD data is better modeled in the multiple instance learning (MIL) framework, and subsequently present a novel convex-hull-based MIL algorithm. In Section 3 we provide experimental evidence from two different CAD problems to show that the proposed algorithm is significantly faster than other MIL algorithms, and more accurate when compared to other MIL algorithms and to traditional classifiers. Further—although this is not the main focus of our paper—on traditional benchmarks for MIL, our algorithm is again shown to be competitive with the current state-of-the-art. We conclude with a description of the relationship to previous work, review of our contributions, and directions for future research in Section 4.

## 2    A Novel Convex Hull MIL algorithm

Almost all the standard classification methods explicitly assume that the training samples (i.e., candidates) are drawn identically and *independently* from an underlying—though unknown—distribution.

This property is clearly violated in a CAD dataset, due to spatial adjacency of the regions identified by a candidate generator, both the features and the class labels of several adjacent candidates (training instances) are highly correlated. First, because the candidate generators for CAD problems are trying to identify potentially suspicious regions, they tend to produce many candidates that are spatially close to each other; since these often refer to regions that are physically adjacent in an image, the class labels for these candidates are also highly correlated. Second, because candidates are labelled positive if they are within some pre-determined distance from a radiologist mark, multiple positive candidates could correspond with the same (positive) radiologist mark on the image. Note that some of the positively labelled candidates may actually refer to healthy structures that just happen to be near a mark, thereby introducing an asymmetric labeling error in the training data.

In MIL terminology from previous literature, a "bag" may contain many observation instances of the same underlying entity, and every training bag is provided a class label (*e.g.* positive or negative). The objective in MIL is to learn a classifier that correctly classifies at least one instance from every bag. This corresponds perfectly with the the appropriate measure of accuracy for evaluating the classifier in a CAD system. In particular, even if one of the candidates that refers to the underlying malignant structure (radiologist mark) is correctly highlighted to the radiologist, the malignant structure is detected; *i.e.* , the correct classification of every candidate instance is not as important as the ability to detect *at least one* candidate that points to a malignant region. Furthermore, we would like to classify every sample that is distant from radiologist mark as negative, this is easily accomplished by considering each negative candidate as a bag. Therefore, it would appear that MIL algorithms should outperform traditional classifiers on CAD datasets.

Unfortunately, in practice, most of the conventional MIL algorithms are computationally quite inefficient, and some of them have problems with local minima. In CAD we typically have several thousand mostly negative candidates (instances), and a few hundred positive bags; existing MIL algorithms are simply unable to handle such large datasets due to time or memory requirements.

**Notation:** Let the $i$-th bag of class $j$ be represented by the matrix $B_j^i \in \Re^{m_j^i \times n}, i = 1, \ldots, r_j$ , $j \in \{\pm 1\}$, $n$ is the number of features. The row $l$ of $B_j^i$, denoted by $B_j^{il}$ represents the datapoint $l$ of the bag $i$ in class $j$ with $l = 1, \ldots, m_j^i$. The binary bag-labels are specified by a vector $d \in \{\pm 1\}^{r_j}$. The vector $e$ represent a vector with all its elements one.

## 2.1 Key idea: Relaxation of MIL via Convex-Hulls

The original MIL problem requires at least one of the samples in a bag to be correctly labeled by the classifier: this corresponds to a set of discrete constraints on the classifier. By contrast, we shall relax this and require that at least one point in the convex hull of a bag of samples (including, possibly one of the original samples) has to be correctly classified. Figure 1 illustrates the idea using a graphical toy example. This relaxation, (first introduced in [1]) eliminates the combinatorial nature of the MIL problem, allowing algorithms that are more computationally efficient. As mentioned above, we will consider that a bag $B_j^i$ is correctly classified if any point inside the convex hull of the bag $B_j^i$ (*i.e.* any convex combination of points of $B_j^i$) is correctly classified. Let $\lambda$ s.t. $0 \le \lambda_j^i, e'\lambda_j^i = 1$ be the vector containing the coefficients of the convex combination that defines the representative point of bag $i$ in class $j$. Let $r$ be the total number of representative points, *i.e.* $r = r_+ + r_-$. Let $\gamma$ be the total number of convex hull coefficients corresponding to the representative points in class $j$, i.e. $\gamma_j = \sum_{i=1}^{r_j} m_j^i$, $\gamma = \gamma_+ + \gamma_-$. Then, we can formulate the MIL problem as,

$$
\begin{aligned}
\min_{(\xi,w,\eta,\lambda)\in R^{r+n+1+\gamma}} \quad & \nu E(\xi) \quad + \quad \Phi(w,\eta) + \ \Psi(\lambda) \\
\text{s.t.} \quad \xi^i \ &= \ d^i - (\lambda_j^i B_j^i w - e\eta) \\
\xi \ &\in \ \Omega \\
e'\lambda_j^i \ &= \ 1 \\
0 \ &\le \ \lambda_j^i
\end{aligned}
\tag{1}
$$

Where $\xi = \{\xi_1, \ldots, \xi_r\}$ are slack terms (errors), $\eta$ is the bias (offset from origin) term, and $\lambda$ is a vector containing all the $\lambda_j^i$ for $i = 1, \ldots, r_j$, $j \in \{\pm\}$. $E : \Re^r \Rightarrow \Re$ represents the loss function, $\Phi : \Re^{(n+1)} \Rightarrow \Re$ is a regularization function on the hyperplane coefficients [2] and $\Psi$ is a regularization function on the convex combination coefficients $\lambda_j^i$. Depending on the choice of $E, \Phi, \Psi$ and $\Omega$, (1) will lead to MIL versions of several well-known classification algorithms.

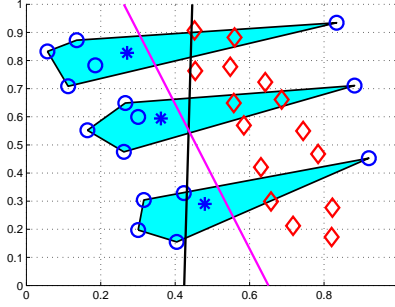

Figure 1: A toy example illustrating the proposed approach. Positive and negative classes are represented by blue circles and red diamonds respectively. Cyan polyhedrons represent the convex hulls for the three positives bags, the points chosen by our algorithm to represent each bag is shown by blue stars. The magenta line represents the linear hyperplane obtained by our algorithm and the black line represents the hyperplane for the SVM.

1. $E(\xi) = \|(\xi)_+\|_2^2$, $\Phi(w, \eta) = \|(w, \eta)\|_2^2$ and $\Omega = \Re^{r+}$, leads to MIL versions of the Quadratic-Programming-SVM [3].

2. $E(\xi) = \|(\xi)\|_2^2$, $\Phi(w, \eta) = \|(w, \eta)\|_2^2$ and $\Omega = \Re^r$, leads to MIL versions of the Least-Squares-SVM.

3. $\nu = 1$, $E(\xi) = \|\xi\|_2^2$, $\Omega = \{\xi : e'\xi_j = 0, \ j \in \{\pm\}\}$ leads to MIL versions of the QP formulation for *Fisher's linear discriminant* (FD) [4].

As an example, we derive a special case of the algorithm for the Fisher's Discriminant, because this choice (FD) brings us some algorithmic as well as computational advantages.

## 2.2 Convex-Hull MIL for Fisher's Linear Discriminant

Setting $\nu = 1$, $E(\xi) = \|\xi\|_2^2$, $\Omega = \{\xi : e'\xi_j = 0, \ j \in \{\pm\}\}$ in (1) we obtain the following MIL version of the quadratic programming algorithm for Fisher's Linear Discriminant [4].

$$
\min_{(\xi, w, \eta, \lambda) \in R^{r+n+1+\gamma}} \quad \|\xi\|_2^2 \quad + \quad \Phi(w, \eta) + \ \Psi(\lambda)
$$
$$
\begin{aligned}
\text{s.t.} \quad \xi^i &= d^i - (\lambda_j^i B_j^i w - e\eta) \\
e'\xi_j &= 0 \\
e'\lambda_j^i &= 1 \\
0 &\leq \lambda_j^i
\end{aligned}
\tag{2}
$$

The number of variables to be optimized in (2) is $r+n+1+\gamma$: this is computationally infeasible when the number of bags is large ($r > 10^4$). To alleviate the situation, we (a) replace $\xi^i$ by $d^i - (\lambda_j^i B_j^i w - e\eta)$ in the objective function, and (b) replace the equality constraints $e'\xi_j = 0$ by $w'(\mu_+ - \mu_-) = 2$. This substitution eliminates the variables $\xi, \eta$ from the problem and also the corresponding $r$ equality constraints in (2). Effectively, this results in the MIL version of the traditional FD algorithm. As discussed later in the paper, in addition to the obvious computational gains, this manipulation results in some algorithmic advantages as well (For more information on the equivalence between the single instance learning versions of (2) and (3) see [4]). Thus, the optimization problem reduces to:

$$
\min_{(w, \ \lambda) \in R^{n+\gamma}} \quad w^T S_W w \quad + \quad \Phi(w) + \ \Psi(\lambda)
$$
$$
\begin{aligned}
\text{s.t.} \quad w^T(\mu_+ - \mu_-) &= b \\
e'\lambda_j^i &= 1 \\
0 &\leq \lambda_j^i
\end{aligned}
\tag{3}
$$

where $S_W = \sum_{j \in \{\pm\}} \frac{1}{r_j}(X_j - \mu_j e')(X_j - \mu_j e')^T$ is the within class scatter matrix, $\mu_j = \frac{1}{r_j}X_j e$ is the mean for class $j$. $X_j \in \Re^{r_j \times n}$ is a matrix containing the $r_j$ representative points on $n$-dimensional space such that the row of $X_j$ denoted by $b_j^i = B_j^i \lambda_j^i$ is the representative point of bag $i$ in class $j$ where $i = \{1, \ldots, r_j\}$ and $j \in \{\pm\}$.

## 2.3 Alternate Optimization for Convex-Hull MIL Fisher's Discriminant

The proposed mathematical program (3) can be solved used an efficient Alternate Optimization (AO) algorithm [5]. In the AO setting the main optimization problem is subdivided in two smaller or easier subproblems that depend on disjoints subsets of the original variables. When $\Phi(w)$ and $\Psi(\lambda)$ are strongly convex functions, both the original objective function and the two subproblems (for optimizing $\lambda$ and $w$) in (3) are strongly convex, meaning that the algorithm converges to a global minimizer [6]. For computational efficiency, in the remainder of the paper we will use the regularizers $\Phi(w) = \epsilon \|w\|_2^2$ and $\Psi(\lambda) = \epsilon \|\lambda\|_2^2$, where $\epsilon$ is a positive regularization parameter. An efficient AO algorithm for solving the mathematical program (3) is described below.

**Sub Problem 1: Fix $\lambda = \lambda^*$:** When we fix $\lambda = \lambda^*$, the problem becomes,

$$\begin{aligned}
\min_{w \in R^n} \quad & w^T S_W w + \Phi(w) \\
\text{s.t.} \quad & w^T (\mu_+ - \mu_-) = b
\end{aligned} \tag{4}$$

which is the formulation for the Fisher's Discriminant. Since $S_W$ is the sum of two covariance matrices, it is guaranteed to be at least positive semidefinite and thus the problem in (4) is convex. For datasets with $r >> n$, i.e. the number of bags is much greater than the number of dimensionality, $S_W$ is positive definite and thus the problem in (4) is strictly convex. Unlike (1) where the number of constraints is proportional to the number of bags, eliminating $\xi$ and $\eta$ leaves us with only one constraint. This changes the order of complexity from $O(nr^2)$ to $O(n^2 r)$ and brings some computational advantages when dealing with datasets with $r >> n$.

**Sub Problem 2: Fix $w = w^*$:** When we fix $w = w^*$, the problem becomes

$$\begin{aligned}
\min_{\lambda \in R^\gamma} \quad & \lambda^T \bar{S}_W \lambda \quad + \quad \Psi(\lambda) \\
\text{s.t.} \quad \lambda^T (\bar{\mu}_+ - \bar{\mu}_-) \ & = \ b \\
e' \lambda_j^i \ & = \ 1 \\
0 \ & \leq \ \lambda_j^i
\end{aligned} \tag{5}$$

where $\bar{S}_W$ and $\bar{\mu}$ are defined as in (4) with $X_j$ replaced by $\bar{X}_j$ where $\bar{X}_j \in \Re^{r_j \times \gamma}$ is now a matrix containing the $r_j$ new points on the $\gamma$-dimensional space such that the row of $\bar{X}_j$ denoted by $\bar{b}_j^i$ is a vector with its nonzero elements set to $B_j^i w^*$. For the positive class elements $\sum_{k=1}^{i-1} m_+^k + 1$ through $\sum_{k=1}^{i} m_+^k$ of $\bar{b}_j^i$ are nonzero, for the negative class nonzero elements are located at $\sum_{k=1}^{r_+} m_+^k + \sum_{k=1}^{i-1} m_-^k + 1$ through $\sum_{k=1}^{r_+} m_+^k + \sum_{k=1}^{i} m_-^k$. Note that $\bar{S}_W$ is also a sum of two covariance matrices, it is positive semidefinite and thus the problem in (5) is convex. Unlike sub problem 1 the positive definiteness of $\bar{S}_W$ does not depend on the data, since it always true that $r \leq \gamma$. The complexity of (5) is $O(n\gamma^2)$.

As it was mentioned before, in CAD applications, a bag is defined as a set of candidates that are spatially close to the radiologist marked ground-truth. Any candidate that is spatially far from this location is considered negative in the training data, therefore the concept of bag for negative examples does not make any practical sense in this scenario. Moreover, since ground truth is only available on the training set, there is no concept of a bag on the test set for both positive and negative examples. The learned classifier labels (ie classifies) individual instances - the bag information for positive examples is only used to help learn a better classifier from the training data. Hence, the problem in (5) can be simplified to account for these practical observations resulting in an optimization problem with $O(n\gamma_+^2)$ complexity. The entire algorithm is summarized below for clarity.

## 2.4 CH-FD: An Algorithm for Learning Convex Hull Representation of Multiple Instances

(0) Choose as initial guess for $\lambda^{i0} = \frac{e}{m^i}$, $\forall i = 1, \ldots, r$, set counter c=0.

(i) For fixed $\lambda^{ic}$, $\forall i = 1, \ldots, r$ solve for $w^c$ in (4).

(ii) Fixing $w = w^c$ solve for $\lambda^{ic}$, $\forall i = 1, \ldots, r$ in (5).

(iii) Stop if $\left\| \lambda^{1(c+1)} - \lambda^{1c}, \ldots, \lambda^{r(c+1)} - \lambda^{rc} \right\|_2$ is less than some desired tolerance. Else replace $\lambda^{ic}$ by $\lambda^{i(c+1)}$ and $c$ by $c + 1$ and go to $(i)$.

The nonlinear version of the proposed algorithm can be obtained by first transforming the original datapoints to a kernel space spanned by all datapoints through a kernel operator, i.e. $K : \Re^n \Rightarrow \Re^{\bar{\gamma}}$ and then by optimizing (4) and (5) in this new space. Ideally $\bar{\gamma}$ is set to $\gamma$. However when $\gamma$ is large, for computational reasons we can use the technique presented in [7] to limit the number of datapoints spanning this new space. This corresponds to constraining $w$ to lie in a subspace of the kernel space.

## 3    Experimental Results and Discussion

For the experiments in section 3.1 , we compare four techniques: naive Fisher's Discriminnat (FD), CH-FD, EM-DD [8], IDAPR [9]. For IDAPR and EM-DD we used the Matlab implementation of these algorithms also used in [10]. In both experiments we used the linear version of our algorithm. Hence the only parameter that requires tuning is $\nu$ which is tuned to optimize the 10-fold Patient Cross Validation on the training data,. All algorithms are trained on the training data and then tested on the sequestered test data. The resulting Receiver Operating Characteristics (ROC) plots are obtained by trying different values of the parameters $(\tau, \epsilon)$ for IDAPR, and by thresholding the corresponding output for each of the EM-DD, FD and CH-FD.

### 3.1    Two CAD Datasets: Pulmonary Embolism & Colon Cancer Detection

Next, we present the problems that mainly motivated this work. Pulmonary embolism (PE), a potentially life-threatening condition, is a result of underlying venous thromboembolic disease. An early and accurate diagnosis is the key to survival. Computed tomography angiography (CTA) has emerged as an accurate diagnostic tool for PE, and However, there are hundreds of CT slices in each CTA study and manual reading is laborious, time consuming and complicated by various PE look-alikes. Several CAD systems are being developed to assist radiologists to detect and characterize emboli [11], [12]. At four different hospitals (two North American sites and two European sites), we collected 72 cases with 242 PE bags comprised of 1069 positive candidates marked by expert chest radiologists. The cases were randomly divided into two sets: training (48 cases with 173 PE bags and 3655 candidates) and testing (24 cases with 69 PE bags and 1857 candidates). The test group was sequestered and only used to evaluate the performance of the final system. A combined total of 70 features are extracted for each candidate.

Colorectal cancer is the third most common cancer in both men and women. It is estimated that in 2004, nearly 147,000 cases of colon and rectal cancer will be diagnosed in the US, and more than 56,730 people would die from colon cancer [13]. CT colonography is emerging as a new procedure to help in early detection of colon polyps. However, reading through a large CT dataset, which typically consists of two CT series of the patient in prone and supine positions, each with several hundred slices, is time-consuming. Colon CAD [14] can play a critical role to help the radiologist avoid the missing of colon polyps. Most polyps, therefore, are represented by two candidates; one obtained from the prone view and the other one from the supine view. Moreover, for large polyps, a typical candidate generation algorithm generates several candidates across the polyp surface. The database of high-resolution CT images used in this study were obtained from seven different sites across US, Europe and Asia. The 188 patients were randomly partitioned into two groups, training comprised of: 65 cases with 127 volumes, 50 polyps bags (179 positive candidates) were identified in this set with a total number of 6569 negative candidates and testing comprised of 123 patients with 237 volumes, a total of 103 polyp bags (232 positive candidates) were identified in this set with a total number of 12752 negative candidates. The test group was sequestered and only used to evaluate the performance of the final system. A total of 75 features are extracted for each candidate.

The resulting Receiver Operating Characteristics (ROC) curves are displayed in Figure 2. Although for the PE dataset Figure 2 (left) IDAPR crosses over CH-FD and is more sensitive than CH-FD for extremely high number of false positives, Table 1 show that CH-FD is more accurate than all other methods over the entire space (AUC). Note that CAD performance is only valid in the clinically acceptable range, $< 10$fp/patient for PE, $< 5$fp/volume for Colon (generally there are 2 volumes per patient). In the region of clinical interest (AUC-RCI), Table 1 shows that CH-FD significantly outperforms all other methods.

Table 1: Comparison of 3 MIL and one traditional algorithms: Computation time, AUC, and normalized AUC in the region of clinical interest for PE and Colon test data

| Algorithm | Time PE | Time Colon | AUC PE | AUC Colon | AUC-RCI PE | AUC-RCI Colon |
|---|---|---|---|---|---|---|
| IAPR | 184.6 | 689.0 | 0.83 | 0.70 | 0.34 | 0.26 |
| EMDD | 903.5 | 16614.0 | 0.67 | 0.80 | 0.17 | 0.42 |
| CH-FD | 97.2 | 7.9 | 0.86 | 0.90 | 0.50 | 0.69 |
| FD | 0.19 | 0.4 | 0.74 | 0.88 | 0.44 | 0.57 |

Execution times for all the methods tested are shown in Table 1. As expected, the computational cost is the cheapest for the traditional non-MIL based FD. Among MIL algorithms, for the PE data, CH-FD was roughly 2-times and 9-times as fast than IAPR and EMDD respectively, and for the much larger colon dataset was roughly 85-times and 2000-times faster, respectively(see Table 1).

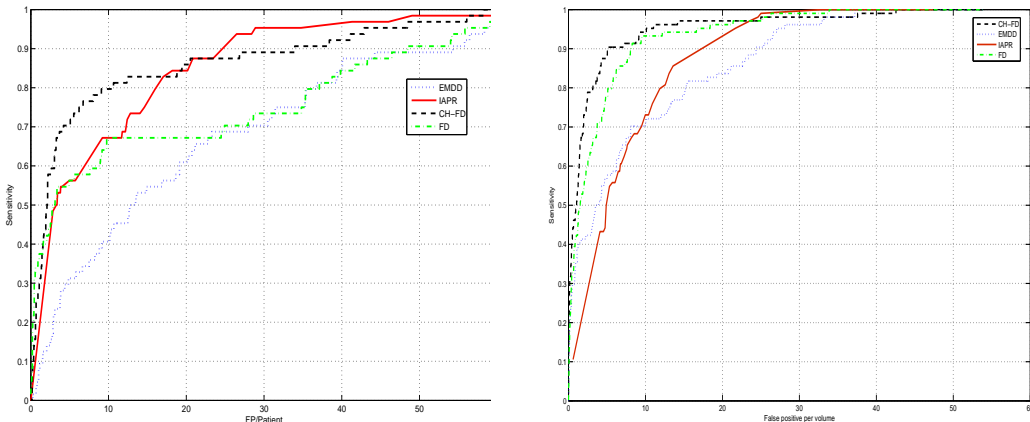

Figure 2: ROC curves obtained for (**left**) PE Testing data and (**right**) COLON testing Data

## 3.2 Experiments on Benchmark Datasets

We compare CH-FD with several state-of-the-art MIL algorithms on 5 benchmark MIL datasets: 2 Musk datasets [9] and 3 Image Annotation datasets [15]. Each of these datasets contain both positive and negative bags. CH-FD (and MICA) use just the positive bag information and ignore the negative bag information, in effect, treating each negative instance as a separate bag. All the other MIL algorithms use both the positive and negative bag information.

The Musk datasets contains feature vectors describing the surfaces of low-energy shapes from molecules. Each feature vector has 166 features. The goal is to differentiate molecules that smell "musky" from the rest of the molecules. Approximately half of the molecules are known to smell musky. There are two musk datasets. MUSK1 contains 92 molecules with a total of 476 instances. MUSK2 contains 102 molecules with a total of 6598 instances. 72 of the molecules are shared between two datasets but MUSK2 dataset contain more instances for the shared molecules.

The Image Annotation data is composed of three different categories, namely *Tiger*, *Elephant*, *Fox*. Each dataset has 100 positive bags and 100 negative bags.

We set $\Phi(w) = \nu |\lambda|$. For the musk datasets our results are based on a Radial Basis Function (RBF) kernel $K(x_i, x_j) = exp(-\sigma \|x - y\|^2)$. The kernel space is assumed to be spanned by all the datapoints in MUSK1 dataset and a subset of the datapoints in MUSK2 dataset (one tenth of the original training set is randomly selected for this purpose). The width of the kernel function and $\nu$ are tuned over a discrete set of five values each to optimize the 10-fold Cross Validation performance. For the Image Annotation data we use the linear version of our algorithm. We follow the benchmark experiment design and report average accuracy of 10 runs of 10-fold Cross Validation

Table 2: Average accuracy on Benchmark Datasets. The number in parenthesis represents the relative rank of each of the algorithms (performance-wise) in the corresponding dataset

| Datasets | MUSK1 | MUSK2 | Elephant | Tiger | Fox | Average Rank |
|----------|-------|-------|----------|-------|-----|--------------|
| CH-FD | 88.8 (2) | 85.7 (2) | 82.4 (2) | 82.2 (2) | 60.4 (2) | 2 |
| IAPR | 87.2 (5) | 83.6 (6) | - (-) | - (-) | - (-) | 5.5 |
| DD | 88.0 (3) | 84.0 (5) | - (-) | - (-) | - (-) | 4 |
| EMDD | 84.8 (6) | 84.9 (3) | 78.3 (5) | 72.1 (5) | 56.1 (5) | 4.8 |
| mi-SVM | 87.4 (4) | 83.6 (6) | 82.2 (3) | 78.4 (4) | 58.2 (3) | 4 |
| MI-SVM | 77.9 (8) | 84.3 (4) | 81.4 (4) | 84.0 (1) | 57.8 (4) | 4.2 |
| MI-NN | 88.9 (1) | 82.5 (7) | - (-) | - (-) | - (-) | 4 |
| MICA | 84.4 (7) | 90.5 (1) | 82.5 (1) | 82.0(3) | 62.0(1) | 3.25 |

in Table 2. Results for other MIL algorithms from the literature are also reported in the same table. Iterated Discriminant APR (IAPR), Diverse Density (DD) [16], Expectation-Maximization Diverse Density (EM-DD) [8], Maximum Bag Margin Formulation of SVM (mi-SVM, MI-SVM) [15], Multi Instance Neural Networks (MI-NN) [17] are the techniques considered in this experiment for comparison purposes. Results for mi-SVM, MI-SVM and EM-DD are taken from [15].

Table 2 shows that CH-FD is comparable to other techniques on all datasets, even though it ignores the negative bag information. Furthermore, CH-FD appears to be the most stable of the algorithms, at least on these 5 datasets, achieving the most consistent performance as indicated by the "Average Rank" column. We believe that this stable behavior of our algorithm is due in part because it converges to global solutions avoiding the local minima problem.

# 4 Discussions

**Relationship to previous literature on MIL:** The Multiple Instance Learning problem described in this paper has been studied widely in the literature [9, 15, 16, 17, 8]. The convex-hull idea presented in this paper to represent each bag is similar in nature to the one presented in [1]. However in contrast with [1] and many other approaches in the literature [9, 15, 17] our formulation leads to a strongly convex minimization problem that converges to a unique minimizer. Since our algorithm considers each negative instance as an individual bag, it is complexity is square proportional to the number of positive instances only which makes it scalable to large datasets with large number of negative examples.

**Principal contributions of the paper:** This paper makes three principal contributions. First, we have identified the need for multiple-instance learning in CAD applications and described the spatial proximity based inter-sample correlations in the label noise for classifier design in this setting. Second, building on an intuitive convex-relaxation of the original MIL problem, this paper presents a new approach to multiple-instance learning. In particular, we dramatically improve the run time by replacing a large set of discrete constraints (at least one instance in each bag has to be correctly classified) with infinite but continuous sets of constraints (at least one convex combination of the original instances in every bag has to be correctly classified). Further, the proposed idea for achieving convexity in the objective function of the training algorithm alleviates the problems of local maxima that occurs in some of the previous MIL algorithms, and often improves the classification accuracy on many practical datasets. Third, we present a practical implementation of this idea in the form of a simple but efficient alternate-optimization algorithm for Convex Hull based Fisher's Discriminant. In our benchmark experiments, the resulting algorithm achieves accuracy that is comparable to the current state of the art, but at a significantly lower run time (typically several orders of magnitude speed ups were observed).

**Related work:** Note that as the distance between candidate ROI increases, the correlations between their features and labels decreases. In another study, we model the spatial-correlation among neighboring samples. Thus we jointly classify entire batches of correlated samples both during training and testing. Instead of classifying each sample independently, we use this spatial information along with the features of each candidate to simultaneously classify all the candidate ROIs for a single patient/volume in a joint operation [18].

# References

[1] O. L. Mangasarian and E. W. Wild. Multiple instance classification via successive linear programming. Technical Report 05-02, Data Mining Institute, Univ of Wisconsin, Madison, 2005.

[2] V. N. Vapnik. *The Nature of Statistical Learning Theory*. Springer, New York, 1995.

[3] O. L. Mangasarian. Generalized support vector machines. In A. Smola, P. Bartlett, B. Schölkopf, and D. Schuurmans, editors, *Advances in Large Margin Classifiers*, pages 135–146, Cambridge, MA, 2000. MIT Press. ftp://ftp.cs.wisc.edu/math-prog/tech-reports/98-14.ps.

[4] Sebastian Mika, Gunnar Rätsch, and Klaus-Robert Müller. A mathematical programming approach to the kernel fisher algorithm. In *NIPS*, pages 591–597, 2000.

[5] J. Bezdek and R. Hathaway. Convergence of alternating optimization. *Neural, Parallel Sci. Comput.*, 11(4):351–368, 2003.

[6] J. Warga. Minimizing certain convex functions. *Journal of SIAM on Applied Mathematics*, 11:588–593, 1963.

[7] Y.-J. Lee and O. L. Mangasarian. RSVM: Reduced support vector machines. Technical Report 00-07, Data Mining Institute, Computer Sciences Department, University of Wisconsin, Madison, Wisconsin, July 2000. Proceedings of the First SIAM International Conference on Data Mining, Chicago, April 5-7, 2001, CD-ROM Proceedings. ftp://ftp.cs.wisc.edu/pub/dmi/tech-reports/00-07.ps.

[8] Q. Zhang and S. Goldman. Em-dd: An improved multiple-instance learning technique. In *Advances in Neural Information Processing Systems*, volume 13. The MIT Press, 2001.

[9] Thomas G. Dietterich, Richard H. Lathrop, and Tomas Lozano-Perez. Solving the multiple instance problem with axis-parallel rectangles. *Artificial Intelligence*, 89(1-2):31–71, 1997.

[10] Z. Zhou and M. Zhang. Ensembles of multi-instance learners. In *Proceedings of the 14th European Conference on Machine Learning, LNAI 2837*, pages 492–502, Cavtat-Dubrovnik, Croatia, 2003. Springer.

[11] M. Quist, H. Bouma, C. Van Kuijk, O. Van Delden, and F. Gerritsen. Computer aided detection of pulmonary embolism on multi-detector ct, 2004.

[12] C. Zhou, L. M. Hadjiiski, B. Sahiner, H.-P. Chan, S. Patel, P. Cascade, E. A. Kazerooni, and J. Wei. Computerized detection of pulmonary embolism in 3D computed tomographic (CT) images: vessel tracking and segmentation techniques. In *Medical Imaging 2003: Image Processing. Edited by Sonka, Milan; Fitzpatrick, J. Michael. Proceedings of the SPIE, Volume 5032, pp. 1613-1620 (2003).*, pages 1613–1620, May 2003.

[13] D. Jemal, R. Tiwari, T. Murray, A. Ghafoor, A. Saumuels, E. Ward, E. Feuer, and M. Thun. Cancer statistics, 2004.

[14] L. Bogoni, P. Cathier, M. Dundar, A. Jerebko, S. Lakare, J. Liang, S. Periaswamy, M. Baker, and M. Macari. Cad for colonography: A tool to address a growing need. *British Journal of Radiology*, 78:57–62, 2005.

[15] S. Andrews, I. Tsochantaridis, and T. Hofmann. Support vector machines for multiple-instance learning. In S. Thrun S. Becker and K. Obermayer, editors, *Advances in Neural Information Processing Systems 15*, pages 561–568. MIT Press, Cambridge, MA, 2003.

[16] Oded Maron and Tomás Lozano-Pérez. A framework for multiple-instance learning. In Michael I. Jordan, Michael J. Kearns, and Sara A. Solla, editors, *Advances in Neural Information Processing Systems*, volume 10. The MIT Press, 1998.

[17] J. Ramon and L. De Raedt. Multi instance neural networks, 2000.

[18] V. Vural, G. Fung, B. Krishnapuram, J. G. Dy, and R. B. Rao. Batch classification with applications in computer aided diagnosis. In *Proceedings of the ECML'06*, Berlin, Germany, 2006.
